# Can V1 mechanisms account for figure-ground and medial axis effects?

**Zhaoping Li**
Gatsby Computational Neuroscience Unit
University College London
zhaoping@gatsby.ucl.ac.uk

## Abstract

When a visual image consists of a figure against a background, V1 cells are physiologically observed to give higher responses to image regions corresponding to the figure relative to their responses to the background. The medial axis of the figure also induces relatively higher responses compared to responses to other locations in the figure (except for the boundary between the figure and the background). Since the receptive fields of V1 cells are very small compared with the global scale of the figure-ground and medial axis effects, it has been suggested that these effects may be caused by feedback from higher visual areas. I show how these effects can be accounted for by V1 mechanisms when the size of the figure is small or is of a certain scale. They are a manifestation of the processes of pre-attentive segmentation which detect and highlight the boundaries between homogeneous image regions.

## 1 Introduction

Segmenting figure from ground is one of the most important visual tasks. We neither know how to execute it on a computer in general, nor do we know how the brain executes it. Further, the medial axis of a figure has been suggested as providing a convenient skeleton representation of its shape (Blum 1973). It is therefore exciting to find that responses of cells in V1, which is usually considered a low level visual area, differentiate between figure and ground (Lamme 1995, Lamme, Zipser, and Spekreijse 1997, Zipser, Lamme, Schiller 1996) and highlight the medial axis (Lee, Mumford, Romero, and Lamme 1998). This happens even though the receptive fields in V1 are much smaller than the scale of these global and perceptually significant phenomena. A common assumption is that feedback from higher visual areas is mainly responsible for these effects. This is supported by the finding that the figure-ground effects in V1 can be strongly reduced or abolished by anaesthesia or lesions in higher visual areas (Lamme et al 1997).

However, in a related experiment (Gallant, van Essen, and Nothdurft 1995), V1 cells were found to give higher responses to global boundaries between two texture regions. Further, this border effect was significant only 10-15 milliseconds after the initial responses of the cells and was present even under anaesthesia. It is thus

plausible that V1 mechanisms is mainly responsible for the border effect.

In this paper, I propose that the figure-ground and medial axis effects are manifestations of the border effect, at least for apropriately sized figures. The border effect is significant within a limited and finite distance from the figure border. Let us call the image region within this finite distance from the border the *effective border region*. When the size of the figure is small enough, all parts of the figure belong to the effective border region and can induce higher responses. This suggests that the figure-ground effect will be reduced or diminished as the size of the figure becomes larger, and the V1 responses to regions of the figure far away from the border will not be significantly higher than responses to background. This suggestion is supported by experimental findings (Lamme et al 1997). Furthermore, the border effect can create secondary ripples as the effect decays with distance from the border. Let us call the distance from the border to the ripple the *ripple wavelength*. When the size of a figure is roughly twice the ripple wavelength, the ripples from the two opposite borders of the figure can reinforce each other at the center of the figure to create the medial axis effect, which, indeed, is observed to occur only for figures of appropriate sizes (Lee et al 1998).

I validate this proposal using a biologically based model of V1 with intra-cortical interactions between cells with nearby but not necessarily overlapping receptive fields. Intra-cortical interactions cause the responses of a cell be modulated by nearby stimuli outside its classical receptive fields — the contextual influences that are observed physiologically (Knierim and van Essen 1992, Kapadia et al 1995). Contextual influences make V1 cells sensitive to global image features, despite their local receptive fields, as manifested in the border and other effects.

## 2   The V1 model

We have previously constructed a V1 model and shown it to be able to highlight smooth contours against a noisy background (Li 1998, 1999, 1999b) and also the boundaries between texture regions in images — the border effect. Its behavior agrees with physiological observations (Knierim and van Essen 1992, Kapadia et al 1995) that the neural response to a bar is suppressed strongly by contextual bars of similar orientatons — iso-orientation suppression; that the response is less suppressed by orthogonally or randomly oriented contextual bars; and that it is enhanced by contextual bars that are aligned to form a smooth contour in which the bar is within the receptive field — contour enhancement. Without loss of generality, the model ignores color, motion, and stereo dimensions, includes mainly layer 2-3 orientation selective cells, and ignores the intra-hypercolumnar mechanism by which their receptive fields are formed. Inputs to the model are images filtered by the edge- or bar-like local receptive fields (RFs) of V1 cells.[1] Cells influence each other contextually via horizontal intra-cortical connections (Rockland and Lund 1983, Gilbert, 1992), transforming patterns of inputs to patterns of cell responses. Fig. 1 shows the elements of the model and their interactions. At each location $i$ there is a model V1 hypercolumn composed of $K$ neuron pairs. Each pair $(i, \theta)$ has RF center $i$ and preferred orientation $\theta = k\pi/K$ for $k = 1, 2, ...K$, and is called (the neural representation of) an edge segment. Based on experimental data (White, 1989), each edge segment consists of an excitatory and an inhibitory neuron that are interconnected, and each model cell represents a collection of local cells of similar types. The excitatory cell receives the visual input; its output is used as a measure of the response or salience of the edge segment and projects to higher visual areas. The inhibitory cells are treated as interneurons. Based on observations

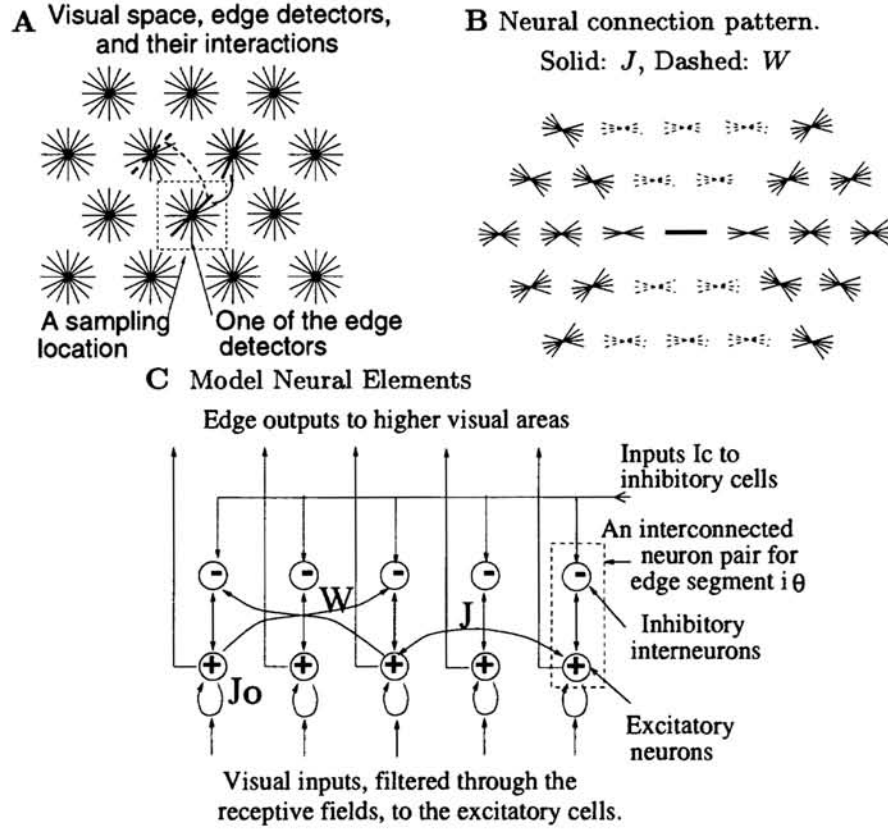

Figure 1: **A:** Visual inputs are sampled in a discrete grid of edge/bar detectors. Each grid point $i$ has $K$ neuron pairs (see **C**), one per bar segment, tuned to different orientations $\theta$ spanning $180°$. Two segments at different grid points can interact with each other via monosynaptic excitation $J$ (the solid arrow from one thick bar to anothe r) or disynaptic inhibition $W$ (the dashed arrow to a thick dashed bar). See also **C**. **B:** A schematic of the neural connection pattern from the center (thick solid) bar to neighboring bars within a few sampling unit distances. $J$'s contacts are shown by thin solid bars. $W$'s are shown by thin dashed bars. The connection pattern is translation and rotation invariant. **C:** An input bar segment is directly processed by an interconnected pair of excitatory and inhibitory cells, each cell models abstractly a local group of cells of the same type. The excitatory cell receives visual input and sends output $g_x(x_{i\theta})$ to higher centers. The inhibitory cell is an interneuron. Visual space is taken as having periodic boundary conditions.

by Gilbert, Lund and their colleagues (Rockland and Lund, 1983, Gilbert 1992) horizontal connections $J_{i\theta,j\theta'}$ (respectively $W_{i\theta,j\theta'}$) mediate contextual influences via monosynaptic excitation (respectively disynaptic inhibition) from $j\theta'$ to $i\theta$ which have nearby but different RF centers, $i \neq j$, and similar orientation preferences, $\theta \sim \theta'$. The membrane potentials follow the equations:

$$\dot{x}_{i\theta} = -\alpha_x x_{i\theta} - \sum_{\Delta\theta} \psi(\Delta\theta) g_y(y_{i,\theta+\Delta\theta}) + J_o g_x(x_{i\theta}) + \sum_{j\neq i,\theta'} J_{i\theta,j\theta'} g_x(x_{j\theta'}) + I_{i\theta} + I_o$$

$$\dot{y}_{i\theta} = -\alpha_y y_{i\theta} + g_x(x_{i\theta}) + \sum_{j\neq i,\theta'} W_{i\theta,j\theta'} g_x(x_{j\theta'}) + I_c$$

where $\alpha_x x_{i\theta}$ and $\alpha_y y_{i\theta}$ model the decay to resting potential, $g_x(x)$ and $g_y(y)$ are sigmoid-like functions modeling cells' firing rates in response to membrane potentials $x$ and $y$, respectively, $\psi(\Delta\theta)$ is the spread of inhibition within a hypercolumn, $J_o g_x(x_{i\theta})$ is self excitation, $I_c$ and $I_o$ are background inputs, including noise and inputs modeling the general and local normalization of activities (see Li (1998) for more details). Visual input $I_{i\theta}$ persists after onset, and initializes the activity levels $g_x(x_{i\theta})$. The activities are then modified by the contextual influences. Depending on the visual input, the system often settles into an oscillatory state (Gray and Singer, 1989, see the details in Li 1998). Temporal averages of $g_x(x_{i\theta})$ over several oscillation cycles are used as the model's output. The nature of the computation performed by the model is determined largely by the horizontal connections $J$ and $W$, which are local (spanning only a few hypercolumns), and translation and rotation invariant (Fig. 1B).

**A:** Input image ($\hat{I}_{i\theta}$) to model          **B:**    Model output

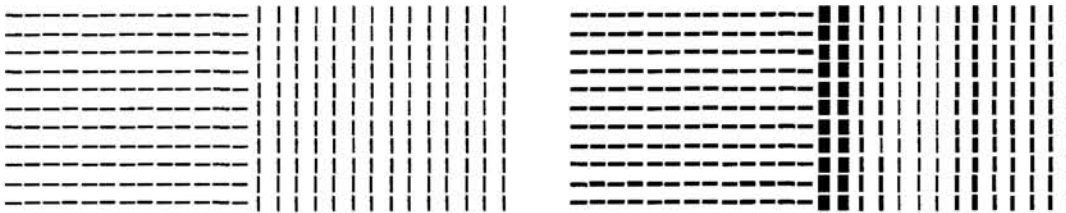

Figure 2: An example of the performance of the model. **A:** Input $\hat{I}_{i\theta}$ consists of two regions; each visible bar has the same input strength. **B:** Model output for **A**, showing non-uniform output strengths (temporal averages of $g_x(x_{i\theta})$) for the edges. The input and output strengths are proportional to the bar widths. Because of the noise in the system, the saliencies of the bars in the same column are not exactly the same, this is also the case in other figures.

The model was applied to some texture border and figure-ground stimuli, as shown in examples in the figures. The input values $\hat{I}_{i\theta}$ are the same for all visible bars in each example. The differences in the outputs are caused by intracortical interactions. They become significant about one membrane time constant after the initial neural response (Li, 1998). The widths of the bars in the figures are proportional to input and output strengths. The plotted region in each picture is often a small region of an extended image. The same model parameters (*e.g.* the dependence of the synaptic weights on distances and orientations, the thresholds and gains in the functions $g_x()$ and $g_y()$, and the level of input noise in $I_o$) are used for all the simulation examples.

Fig. 2 demonstrates that the model indeed gives higher responses to the boundaries between texture regions. This border effect is highly significant within a distance of about 2 texture element spacings from the border. Thus the effective border region is about 2 in texture element spacings in this example. Furthermore, at about 9 texture element spacings to the right of the texture border there is a much smaller but significant (visible on the figure) secondary peak in the response amplitude. Thus the ripple wavelength is about 9 texture element spacings here. The border effect is mainly caused by the fact that the texture elements at the border experience less iso-orientation suppression (which reduces the response levels to other texture bars in the middle of a homogeneous (texture) region) — the texture elements at the border have fewer neighboring texture bars of a similar orientation than the texture elements in the centers of the regions. The stronger responses to the effective border region cause extra iso-orientation suppression to texture bars near but right outside the effective border region. Let us call this region of stronger

**Model Input**          **Model Output**

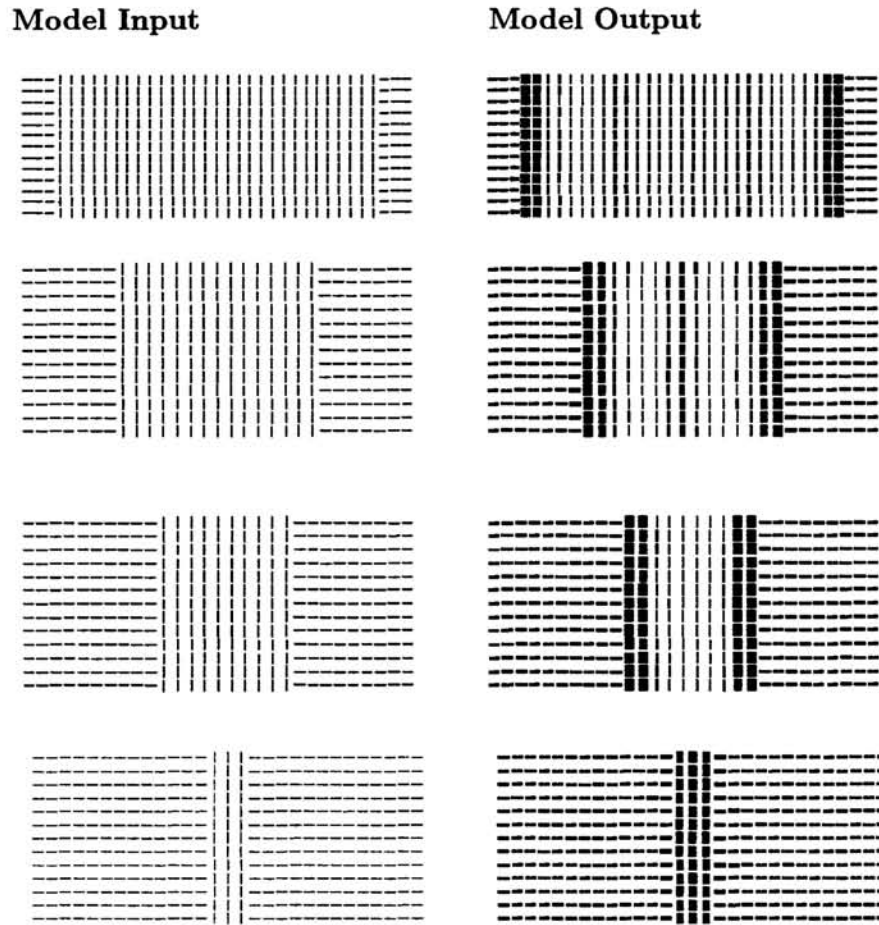

Figure 3: Dependence on the size of the figure. The figure-ground effect is most evident only for small figures, and the medial axis effect is most evident only for figures of finite and appropriate sizes.

suppression from the border the *border suppression region*, which is significant and visible in Fig. (2B). This region can reach no further than the longest length of the horizontal connenctions (mediating the suppresion) from the effective border region. Consequently, texture bars right outside the border suppression region not only escape the stronger suppression from the border, but also experience weaker iso-orientation suppression from the weakened texture bars in the nearby border suppression region. As a result, a second saliency peak appears — the ripple effect, and we can hence conclude that the ripple wavelength is of the same order of magnitude as the longest connection length of the cortical lateral connections mediating intra-cortical interactions.

Fig. 3 shows that for very small figures, the whole figure belongs to the effective border region and is highlighted in the V1 responses. As the figure size increases, the responses in the inside of the figure become smaller than the responses in the border region. However, when the size of the figure is appropriate, namely about twice the ripple wavelength, the center of the figure induces a secondary response highlight. In this case, the ripples or the secondary saliency peaks from both borders superpose onto each other at the same spatial location at the center of the figure. This reinforces the saliency peak at this medial axis since it has two border suppression regions (from two opposite borders), one on each side of it, as its contextual stimuli. For even larger figures, the medial axis effect diminishes because the ripples from

**Model Input**          **Model Output**

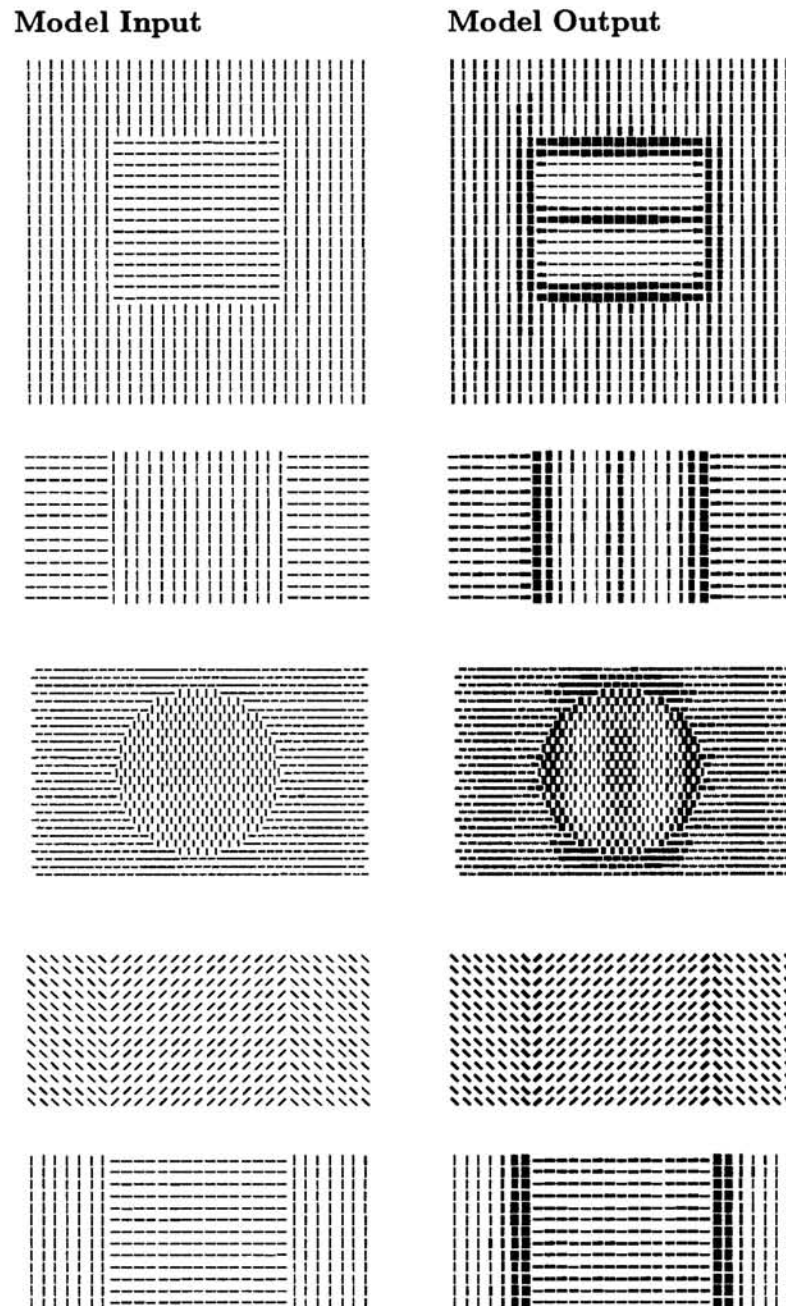

Figure 4: Dependence on the shape and texture feature of the figures.

the two opposite borders of the figure no longer reinforce each other.

Fig. 4 demonstrates that the border effect and its consequences for the medial axis also depend on the shape of the figures and the nature of the texture they contain (*eg* the orientations of the elements). Bars in the texture parallel to the border induce stronger highlights, and as a consequence, cause stronger ripple effects and medial axis highlights. This comes from the stronger co-linear, contour enhancing, inputs these bars receive than bars not parallel to the border.

## 3   Summary and Discussion

The model of V1 was originally proposed to account for pre-attentive contour enhancement and visual segmentation (Li 1998, 1999, 1999b). The contextual influences mediated by intracortical interactions enable each V1 neuron to process inputs from a local image area substantially larger than its classical receptive field. This enables cortical neurons to detect image locations where translation invariance in the input image breaks down, and highlight these image locations with higher neural activities, making them conspicuous. These highlights mark candidate locations for image region (or object surface) boundaries, smooth contours and small figures against backgrounds, serving the purpose of pre-attentive segmentation.

This paper has shown that the figure-ground and medial axis effects observed in the recent experiments can be accounted for using a purely V1 mechanism for border highlighting, provided that the sizes of the figures are small enough or of finite and appropriate scale. This has been the case in the existing experiments. We therefore suggest that feedbacks from higher visual areas are not necessary to explain the experimental observations, although we cannot, of course, exclude the possibilities that they also contribute.

## Footnotes

[1]The terms 'edge' and 'bar' will be used interchangeably.

## References

[1] Lamme V.A. (1995) *Journal of Neuroscience* **15**(2), 1605-15.

[2] Lee T.S, Mumford D, Romero R. and Lamme V. A.F. (1998) *Vis. Res.* 38: 2429-2454.

[3] Zipser K., Lamme V. A., and Schiller P. H. (1996) *J. Neurosci.* **16** (22), 7376-89.

[4] Lamme V. A. F., Zipser K. and Spekreijse H. *Soc. Neuroscience Abstract* 603.1, 1997.

[5] Blum H. (1973) Biological shape and visual science *J. Theor. Biol.* 38: 205-87.

[6] Gallant J.L., van Essen D.C., and Nothdurft H.C. (1995) In *Early vision and beyond* eds. T. Papathomas, Chubb C, Gorea A., and Kowler E. (MIT press), pp 89-98.

[7] C. D. Gilbert (1992) *Neuron.* **9**(1), 1-13.

[8] C. M. Gray and W. Singer (1989) *Proc. Natl. Acad. Sci. USA* **86**, 1698-1702.

[9] M. K. Kapadia, M. Ito, C. D. Gilbert, and G. Westheimer (1995) *Neuron.* **15**(4), 843-56.

[10] J. J. Knierim and D. C. van Essen (1992) *J. Neurophysiol.* **67**, 961-980.

[11] Z. Li (1998) *Neural Computation* 10(4) p 903-940.

[12] Z. Li (1999) *Network: computations in neural systems* 10(2). p. 187-212.

[13] Z. Li (1999b) *Spatial Vision* 13(1) p. 25-50.

[14] K.S. Rockland and J. S. Lund (1983) *J. Comp. Neurol.* **216**, 303-318

[15] E. L. White (1989) *Cortical circuits* (Birkhauser).
